# Effective Dimension and Generalization of Kernel Learning

**Tong Zhang**
IBM T.J. Watson Research Center
Yorktown Heights, NY 10598
tzhang@watson.ibm.com

## Abstract

We investigate the generalization performance of some learning problems in Hilbert function Spaces. We introduce a concept of scale-sensitive effective data dimension, and show that it characterizes the convergence rate of the underlying learning problem. Using this concept, we can naturally extend results for parametric estimation problems in finite dimensional spaces to non-parametric kernel learning methods. We derive upper bounds on the generalization performance and show that the resulting convergent rates are optimal under various circumstances.

## 1 Introduction

The goal of supervised learning is to predict an unobserved output value $y$ based on an observed input vector $x$. This requires us to estimate a functional relationship $y \approx p(x)$ from a set of training examples. Usually the quality of the predictor $p(x)$ can be measured by a loss function $f(p(x), y)$. In machine learning, we assume that the data $(x, y)$ are drawn from an unknown underlying distribution. Our goal is to find $p(x)$ so that the expected true loss of $p$ given below is as small as possible:

$$L(p(\cdot)) = E_{x,y} f(p(x), y),$$

where we use $E_{x,y}$ to denote the expectation with respect to the true (but unknown) underlying distribution.

In this paper we focus on smooth convex loss functions $f$ that are second order differentiable with respect to the first component. In addition we assume that the second derivative is bounded both above and below (away from zero).[1] For example, our analysis applies to important methods such as least squares regression (aka, Gaussian processes) and logistic regression in Hilbert spaces.

In order to obtain a good predictor $\hat{p}(x)$ from training data, it is necessary to start with a model of the functional relationship. In this paper, we consider models that are subsets in some Hilbert function space $H$. Denote by $\| \cdot \|_H$ the norm in $H$. In particular, we consider models in a bounded convex subset $\Omega$ of $H$. We would like to find the best model in $\Omega$

defined as:

$$p_\Omega(\cdot) = \arg\min_{p \in \Omega} L(p) = \arg\min_{p \in \Omega} E_{x,y} f(p(x), y). \qquad (1)$$

In supervised learning, we construct an estimator $\hat{p}$ of $p_\Omega(\cdot)$ from a set of $n$ training examples $S = \{(x_1, y_1), \ldots, (x_n, y_n)\}$. Throughout the paper, we use symbol $\hat{\ }$ to denote empirical quantities based on the $n$ observed training data $S$. Specifically, we use $\hat{E}_{x,y}$ to denote the empirical expectation with respect to the training samples, and

$$\hat{L}(p) = \hat{E}_{x,y} f(p(x), y) = \frac{1}{n} \sum_{i=1}^{n} f(p(x_i), y_i).$$

Assume that input $x$ belongs to a set $X$. We make the reasonable assumption that $p$ is point-wise continuous under the $\|\cdot\|_H$ topology: $\forall x \in X$, $\lim_{p \to p_0} p(x) = p_0(x)$ where $p \to p_0$ is in the sense that $\|p - p_0\|_H \to 0$. This assumption is equivalent to the condition $\sup_{\|p\|_H \le 1} p(x) < +\infty$ ($\forall x \in X$), implying that each data point $x$ can be regarded as a bounded linear functional $\phi_x$ on $H$ such that $\forall p \in H$: $\phi_x(p) = p(x)$. Since a Hilbert space $H$ is self-dual, we can represent $\phi_x$ by an element in $H$. Therefore $\forall x$ we can define $\phi_x \in H$ as $\phi_x \cdot p = p(x)$ for all $p \in H$, where $\cdot$ denotes the inner product of $H$.

It is clear that $\phi_x$ can be regarded as a representing feature vector of $x$ in $H$. In the literature, the inner product $K(x_1, x_2) = \phi_{x_1} \cdot \phi_{x_2}$ is often referred to as the kernel of $H$, and $H$ as the reproducing kernel Hilbert space which is determined by the kernel function $K(x_1, x_2)$.

The purpose of this paper is to develop bounds on the true risk $L(\hat{p})$ of any empirical estimator $\hat{p}$ compared to the optimal risk $L(p_\Omega)$ based on its observed risk $\hat{L}(\hat{p})$. Specifically we seek a bound of the following form:

$$L(\hat{p}) \le L(p_\Omega) + c(\hat{L}(\hat{p}) - \hat{L}(p_\Omega)) + \Delta(\hat{p}, \lambda, n),$$

where $c$ is a positive constant that only depends on the loss function $f$, and $\lambda$ is a parameter that characterizes the effective data dimensionality for the learning problem.

If $\hat{p}$ is the empirical estimator that minimizes $\hat{L}$ in $\Omega$, then the second term on the right hand side is non-positive. We are thus mainly interested in the third term. It will be shown that if $H$ is a finite dimensional space, then the third term is $O(d/n)$ where $d = \dim(H)$ is the dimension of $H$. If $H$ is an infinite dimensional space (or when $d$ is large compared to $n$), one can adjust $\lambda$ appropriately based on the sample size $n$ to get a bound $O(d_n/n)$ where the effective dimension $d_n$ at the optimal scale $\lambda$ becomes sample-size dependent. However the dimension will never grow faster than $d_n = O(\sqrt{n})$ and hence even in the worse case, $\Delta(\hat{p}, \lambda, n)$ converges to zero at a rate no worse than $O(1/\sqrt{n})$.

A consequence of our analysis is to obtain convergence rates better than $O(1/\sqrt{n})$. For empirical estimators with least squares loss, this issue has been considered in [1, 2, 4] among others. The approach in [1] won't lead to the optimal rate of convergence for non-parametric classes. The $L_2$-covering number based analysis in [2, 4] use the chaining argument [4] and ratio large deviation inequalities. However, it is known that chaining does not always lead to the optimal convergence rate, and for many problems covering numbers can be rather difficult to estimate. The effective dimension based analysis presented here, while restricted to learning problems in Hilbert spaces (kernel methods), addresses these issues.

## 2   Decomposition of loss function

Consider a convex subset $\Omega \subset H$, which is closed under the uniform norm topology. Let $p_\Omega$ be the optimal predictor $p_\Omega$ in $\Omega$ defined in (1). By differentiating (1) at the optimal

solution, and using the convexity of $\Omega$ with respect to $p$, we obtain the following first order condition:

$$E_{x,y}f_1'(p_\Omega(x),y)(p(x) - p_\Omega(x)) \geq 0 \qquad (\forall p \in \Omega), \tag{2}$$

where $f_1'(p,y)$ is the derivative of $f(p,y)$ with respect to $p$. This inequality will be very important in our analysis.

**Definition 2.1** *The Bregman distance of $f$ (with respect to its first variable) is defined as:*

$$d_f(p,q;y) = f(q,y) - f(p,y) - f_1'(p,y)(q-p).$$

It is well known (and easy to check) that for a convex function, its Bregman divergence is always non-negative. As mentioned in the introduction, we assume for simplicity that there exist positive constants $c_l$ and $c_u$ such that $0 < c_l \leq f_1''(p,y)/2 \leq c_u$, where $f_1''$ is the second order derivative of $f$ with respect to the first variable. Using Taylor expansion of $f$, it is easy to see that we have the following inequality for $d_f$:

$$c_l(p-q)^2 \leq d_f(p,q;y) \leq c_u(p-q)^2. \tag{3}$$

Now, $\forall f \in \Omega$, we consider the following decomposition:

$$L(p) - L(p_\Omega) = E_{x,y}d_f(p_\Omega(x),p(x);y) + E_{x,y}f_1'(p_\Omega(x),y)(p(x) - p_\Omega(x)).$$

Clearly by the non-negativeness of Bregman divergence and (2), the two terms on the right hand side of the above equality are all non-negative. This fact is very important in our approach. The above decomposition of $L$ gives the following decomposition of loss function:

$$f(p(x),y) - f(p_\Omega(x),y) = d_f(p_\Omega(x),p(x);y) + f_1'(p_\Omega(x),y)(p(x) - p_\Omega(x)).$$

We thus obtain from (3):

$$\begin{aligned} &c_l(p(x) - p_\Omega(x))^2 + f_1'(p_\Omega(x),y)(p(x) - p_\Omega(x)) \\ \leq &f(p(x),y) - f(p_\Omega(x),y) \\ \leq &c_u(p(x) - p_\Omega(x))^2 + f_1'(p_\Omega(x),y)(p(x) - p_\Omega(x)). \end{aligned} \tag{4}$$

## 3   Empirical ratio inequality and generalization bounds

Given a positive definite self-adjoint operator $Q : H \to H$, we define an inner product structure on $H$ as:

$$\langle u,v \rangle_Q = u \cdot Qv = u^T Qv.$$

The corresponding norm is $\|u\|_Q = \langle u,u \rangle_Q^{1/2}$.

Given a positive number $\lambda$, and let $I$ be the identity operator, we define the following self-adjoint operator on $H$:

$$Q_\lambda = (E_x \phi_x \phi_x^T + \lambda I)^{-1},$$

where we have used the matrix notation $\phi_x \phi_x^T$ to denote the self-adjoint operator $H \to H$ defined as: $(\phi_x \phi_x^T)h = \phi_x(\phi_x \cdot h) = h(x)\phi_x$.

In addition, we consider the inner product space $T_\lambda$ on the set of self-adjoint operators on $H$, with the inner product defined as

$$\langle A,B \rangle_{T_\lambda} = \mathrm{tr}(Q_\lambda A Q_\lambda B),$$

where $\mathrm{tr}(S)$ is the trace of a linear operator $S$ (sum of eigenvalues). The corresponding norm is denoted as $\| \cdot \|_{T_\lambda}$.

We start our analysis with the following simple lemma:

**Lemma 3.1** *For any function $a(x, y)$, the following bounds are valid:*

$$\sup_{p \in H} \frac{|\hat{E}_{x,y} a(x,y) p(x) - E_{x,y} a(x,y) p(x)|}{\sqrt{E_x p(x)^2 + \lambda \|p\|_H^2}} \le \|\hat{E}_{x,y} a(x,y) \phi_x - E_{x,y} a(x,y) \phi_x\|_{Q_\lambda},$$

$$\sup_{p \in H} \frac{|\hat{E}_x p(x)^2 - E_x p(x)^2|}{E_x p(x)^2 + \lambda \|p\|_H^2} \le \|\hat{E}_x \phi_x \phi_x^T - E_x \phi_x \phi_x^T\|_{T_\lambda}.$$

**Proof** Note that $E_x p(x)^2 + \lambda \|p\|_H^2 = p^T Q_\lambda^{-1} p$. Therefore let $v = \hat{E}_{x,y} a(x,y) \phi_x - E_{x,y} a(x,y) \phi_x$, we obtain from Cauchy-Schwartz inequality

$$|\hat{E}_{x,y} a(x,y) p(x) - E_{x,y} a(x,y) p(x)| = |p \cdot v| \le (p^T Q_\lambda^{-1} p)^{1/2} (v^T Q_\lambda v)^{1/2}.$$

This proves the first inequality.

To show the second inequality, we simply observe that the left hand side is the largest absolute eigenvalue of the operator $A = Q_\lambda (\hat{E}_x \phi_x \phi_x^T - E_x \phi_x \phi_x^T)$, which is upper bounded by $\sqrt{\operatorname{tr}(A^2)}$. Therefore the second inequality follows immediately from the definition of $T_\lambda$-norm. □

The importance of Lemma 3.1 is that it bounds the behavior of any estimator $p \in H$ (which can be sample dependent) in terms of the norm of the empirical mean of $n$ zero-mean Hilbert-space valued random vectors. The convergence rate of the latter can be easily estimated from the variance of the random vectors, and therefore we have significantly simplified the problem.

In order to estimate the variance of the random vectors on the right hand sides of Lemma 3.1, and hence characterize the behavior of the learning problem, we shall introduce the following notion of effective data dimensionality at a scale $\lambda$:

$$D_\lambda = E_x \phi_x^T Q_\lambda \phi_x = E_x \|\phi_x\|_{Q_\lambda}^2.$$

Some properties of $D_\lambda$ are listed in Appendix A, which can be used to estimate the quantity. In particular for a finite dimensional space $H$, $D_\lambda$ is upper bounded by the dimensionality $\dim(H)$ of the space. Moreover the equality can be achieved by letting $\lambda \to 0$ as long as $E_x \phi_x \phi_x^T$ is full rank. Thus this quantity behaves like (scale-sensitive) data dimension.

We also define the following quantities to measure the boundedness of the input data:

$$M_H \ge \sup_x \|\phi_x\|_H, \qquad M_\lambda = \sup_x \|\phi_x\|_{Q_\lambda}. \tag{5}$$

It is easy to see that $M_\lambda \le M_H / \sqrt{\lambda}$.

**Lemma 3.2** *Let $c = \sup_{x,y} a(x,y)$, then we have*

$$E_{x,y} \|a(x,y) \phi_x - E_{x',y'} a(x',y') \phi_{x'}\|_{Q_\lambda}^2 \le c^2 D_\lambda,$$

$$E_x \|\phi_x \phi_x^T\|_{T_\lambda} = D_\lambda, \quad E_x \|\phi_x \phi_x^T - E_{x'} \phi_{x'} \phi_{x'}^T\|_{T_\lambda}^2 \le D_\lambda M_\lambda^2.$$

**Proof** Let $\phi = E_{x',y'} a(x',y') \phi_{x'}$, then we have

$$E_{x,y} \|a(x,y) \phi_x - \phi\|_{Q_\lambda}^2 = E_{x,y} \|a(x,y) \phi_x\|_{Q_\lambda}^2 - \|\phi\|_{Q_\lambda}^2 \le c^2 D_\lambda,$$

which gives the first inequality.

Note that $\forall \phi \in H: \|\phi \phi^T\|_{T_\lambda} = \|\phi\|_{Q_\lambda}^2$. Therefore

$$E_x \|\phi_x \phi_x^T\|_{T_\lambda} = E_x \|\phi_x\|_{Q_\lambda}^2 = D_\lambda,$$

leading to the second equality. Since $\|\phi_x\phi_x^T\|_{T_\lambda} = \|\phi_x\|_{Q_\lambda}^2 \le M_\lambda{}^2$, we have

$$E_x \|\phi_x\phi_x^T\|_{T_\lambda}^2 \le E_x \|\phi_x\phi_x^T\| M_\lambda{}^2 \le D_\lambda M_\lambda{}^2.$$

Similar to the proof of the first inequality, it is easy to check that this implies the third inequality. $\square$

Next we need to use the following version of Bernstein inequality in Hilbert spaces.

**Proposition 3.1 ([5])** *Let $\xi_i$ be zero-mean independent random vectors in a Hilbert space. If there exist $B, M > 0$ such that for all natural numbers $l \ge 2$: $\frac{1}{n}\sum_{i=1}^n E\|\xi_i\|_H^l \le \frac{B^2}{2}l!M^{l-2}$. Then for all $\delta > 0$: $P(\|\frac{1}{n}\sum_i \xi_i\|_H \ge \delta) \le 2\exp(-\frac{n}{2}\delta^2/(B^2 + \delta M))$.*

In this paper, we shall use the following variant of the above bound for convenience.

$$P\left(\left\|\frac{1}{n}\sum_i \xi_i\right\|_H \ge \frac{2Mt}{n} + \sqrt{\frac{2t}{n}}B\right) \le 2\exp(-t). \tag{6}$$

**Lemma 3.3** *Under the assumptions of Lemma 3.2, let $\epsilon_\lambda(t) = \sqrt{\frac{2tD_\lambda}{n}} + \frac{4tM_\lambda}{n}$. Then with probability of at least $1 - 2\exp(-t)$:*

$$\sup_{p\in H} \frac{|\hat{E}_{x,y}a(x,y)p(x) - E_{x,y}a(x,y)p(x)|}{\sqrt{E_x p(x)^2 + \lambda\|p\|_H^2}} \le \epsilon_\lambda(t)c.$$

*Similarly, with probability of at least $1 - 2\exp(-t)$, we have:*

$$\sup_{p\in H} \frac{|\hat{E}_x p(x)^2 - E_x p(x)^2|}{E_x p(x)^2 + \lambda\|p\|_H^2} \le \epsilon_\lambda(t)M_\lambda.$$

**Proof** The bounds are straight forward applications of (6) and the previous two lemmas. Due to the limitation of space, we skip the details. $\square$

We are now ready to derive the following main result of the paper:

**Theorem 3.1** *Assume $\sup_{x,y}|f_1'(p_\Omega(x),y)| \le c_\Omega$. Let $\alpha = c_u/c_l$ where $c_l$ and $c_u$ satisfy (3). Consider any sample dependent estimator $\hat{p}$ such that $\hat{p} \in \Omega$. That is, $\hat{p} \in \Omega$ is a function of the training sample $S$. Let $\epsilon_\lambda(t) = \sqrt{\frac{2tD_\lambda}{n}} + \frac{4tM_\lambda}{n}$. If we choose $\lambda$ such that $\alpha\epsilon_\lambda(t)M_\lambda \le 0.5$, then with probability of at least $1 - 4\exp(-t)$, the generalization error is bounded as:*

$$L(\hat{p}) \le L(p_\Omega) + 4\alpha[\hat{L}(\hat{p}) - \hat{L}(p_\Omega)] + 3\lambda c_l\|\hat{p} - p_\Omega\|_H^2 + \frac{4\alpha^2 c_\Omega^2 \epsilon_\lambda(t)^2}{c_l}.$$

**Proof** We introduce the following notations for convenience:

$$\hat{A}(p) = \hat{E}_{x,y}f_1'(p_\Omega(x),y)(p(x) - p_\Omega(x)), \quad A(p) = E_{x,y}f_1'(p_\Omega(x),y)(p(x) - p_\Omega(x)),$$
$$\hat{B}(p) = \hat{E}_x(p(x) - p_\Omega(x))^2, \quad B(p) = E_x(p(x) - p_\Omega(x))^2,$$
$$C(p) = B(p) + \lambda\|p - p_\Omega\|_H^2.$$

We obtain from Lemma 3.3 that with probability of at least $1 - 4\exp(-t)$:

$$|\hat{A}(\hat{p}) - A(\hat{p})| \le \epsilon_\lambda(t)c_\Omega C(\hat{p})^{1/2}, \qquad |\hat{B}(\hat{p}) - B(\hat{p})| \le \epsilon_\lambda(t)M_\lambda C(\hat{p}).$$

Combining the above two inequalities, we obtain:

$$|\hat{A}(\hat{p}) - A(\hat{p})| + c_l|\hat{B}(\hat{p}) - B(\hat{p})| \le \epsilon_\lambda(t)[c_\Omega C(\hat{p})^{1/2} + c_l M_\lambda C(\hat{p})].$$

Using (4) and recalling (2), we obtain

$$\frac{c_l}{c_u}[L(\hat{p}) - L(p_\Omega)] \le [\hat{L}(\hat{p}) - \hat{L}(p_\Omega)] + \epsilon_\lambda(t)[c_\Omega C(\hat{p})^{1/2} + c_l M_\lambda C(\hat{p})]. \qquad (7)$$

Let

$$K_1(p) = [L(p) - L(p_\Omega)] + \lambda c_l \|p - p_\Omega\|_H^2, \quad \hat{K}_2(p) = [\hat{L}(p) - \hat{L}(p_\Omega)] + \lambda \frac{c_l^2}{c_u} \|p - p_\Omega\|_H^2,$$

then (2) and (4) imply that $c_l C(p) \le K_1(p)$. We can derive from (7)

$$\frac{c_l}{c_u} K_1(\hat{p}) \le \hat{K}_2(\hat{p}) + \epsilon_\lambda(t) \left[ c_\Omega \sqrt{\frac{K_1(\hat{p})}{c_l}} + M_\lambda K_1(\hat{p}) \right].$$

Using the assumption that $\alpha \epsilon_\lambda(t) M_\lambda \le 0.5$, we obtain

$$\frac{c_l}{c_u} K_1(\hat{p}) \le 2\hat{K}_2(\hat{p}) + 2\epsilon_\lambda(t) c_\Omega \sqrt{\frac{K_1(\hat{p})}{c_l}},$$

which can be regarded as a quadratic inequality of $K_1^{1/2}(\hat{p})$. Solving the inequality using elementary algebra, we obtain:

$$K_1(\hat{p}) \le 4\alpha \hat{K}_2(\hat{p}) + \frac{4\alpha^2 c_\Omega^2 \epsilon_\lambda(t)^2}{c_l},$$

which immediately implies the theorem. □

Note that both $D_\lambda$ and $M_\lambda$ go to zero as $\lambda \to \infty$, therefore the assumption $\alpha \epsilon_\lambda(t) M_\lambda \le 0.5$ can be satisfied as long as we pick $\lambda$ that is larger than a critical value $\lambda_0$. Using the bound $M_\lambda \le M_H/\sqrt{\lambda}$, we easily obtain the following result.

**Corollary 3.1** *Under the assumptions of Theorem 3.1. Assume also that the diameter of $\Omega$ is bounded by $A$: $A = \sup_{p,q \in \Omega} \|p - q\|_H$. Then for all $\lambda$ and an upper bound $\tilde{D}_\lambda$ of $D_\lambda$. If $2\alpha \tilde{D}_\lambda \ge 1$ and $\lambda/\tilde{D}_\lambda \ge 32\alpha^2 t M_H^2/n$, we have with probability of at least $1 - 4\exp(-t)$,*

$$L(\hat{p}) \le L(p_\Omega) + 4\alpha[\hat{L}(\hat{p}) - \hat{L}(p_\Omega)] + \lambda(3A^2 c_l + \frac{c_\Omega^2}{c_l M_H^2}).$$

## 4 Examples

We will only consider empirical estimator $\hat{p}$ that minimizes $\hat{L}(p)$ in $\Omega$. In this case, $[\hat{L}(\hat{p}) - \hat{L}(p_\Omega)] \le 0$ in Corollary 3.1. We shall thus only focus on the third term.

**Worst case effective dimensionality and generalization**

In the worst case, we have $D_\lambda \le M_H^2/\lambda$. Therefore if $8t \le n$, we can always let $\lambda = 4\sqrt{2}\alpha M_H^2 \sqrt{t/n}$ in Corollary 3.1 and obtain with probability at least $1 - 4\exp(-t)$:

$$L(\hat{p}) \le L(p_\Omega) + 4\alpha(3c_l A^2 M_H^2 + \frac{c_\Omega^2}{c_l})\sqrt{\frac{2t}{n}}.$$

**Finite dimensional problems**

We can use the bound $D_\lambda \le \dim(H)$. Therefore we can let $\lambda = 32 \dim(H)\alpha^2 t M_H^2/n$ in Corollary 3.1 and obtain:

$$L(\hat{p}) \le L(p_\Omega) + 32 \dim(H)\alpha^2 M_H^2 (3c_l A^2 M_H^2 + \frac{c_\Omega^2}{c_l})\frac{t}{n}.$$

It is well known that the rate of the order $O(\dim(H)/n)$ is optimal in this case.

**Smoothing splines**

For simplicity, we only consider 1-dimensional problems. For smoothing splines, the corresponding Hilbert space consists of functions $p$ satisfying the smoothness condition that $\int [p^{(s)}(x)]^2 dx$ is bounded ($p^{(s)}$ is the $s$-th derivative of $p$ and $s > 1/2$). We may consider periodic functions (or their restrictions in an interval) and the condition corresponds to a decaying Fourier coefficients condition. Specifically, the space can be regarded as the reproducing kernel Hilbert space with kernel

$$K(x_1, x_2) = \sum_{k \ge 0} (k+1)^{-2s}(\sin(kx_1)\sin(kx_2) + \cos(kx_1)\cos(kx_2)).$$

Now, using Proposition A.3, we have $D_\lambda \le \inf_{k \ge 1}\left[2k + \frac{2/\lambda}{(2s-1)k^{2s-1}}\right]$. Therefore $D_{k^{-2s}} \le \frac{4sk}{2s-1}$. Note that we may take $M_H^2 = 2/(2s-1)$. Therefore assuming $(2s-1)^2 n \ge 2^{2s+10}s\alpha^2 t$ we can let $\lambda = k^{-2s}$ in Corollary 3.1 where $k$ is the largest integer such that $k^{2s+1} \le \frac{(2s-1)^2 n}{2^9 s\alpha^2 t}$. This gives the following bound (with probability at least $1 - 4\exp(-t)$).

$$L(\hat{p}) \le L(p_\Omega) + 2^{2s}(\frac{6c_l A^2}{2s-1} + \frac{c_\Omega^2}{c_l})\left(\frac{2^9 s\alpha^2 t}{(2s-1)^2 n}\right)^{2s/(2s+1)}.$$

This rate matches the best possible convergence rate for any data-dependent estimator.[2]

**Exponential kernel**

Exponential kernel has recently been popularized by Vapnik. Again for simplicity we consider 1-dimensional problems where $x \in [-1, 1]$. The kernel function is given by

$$K(x_1, x_2) = \exp(x_1 x_2) = \sum_{i=0}^{n} \frac{1}{i!}x_1^i x_2^i.$$

Therefore $D_\lambda \le \inf_{k \ge 0}[k + 2 + \frac{1}{\lambda k!}]$. We obtain an upper bound $L(\hat{p}) \le L(p_\Omega) + O(\frac{\ln n}{n \ln \ln n})$, implying that the effective dimension is at most $O(\ln n/\ln \ln n)$ for exponential kernels.

## 5 Conclusion

In this paper, we introduced a concept of scale-sensitive effective data dimension, and used it to derive generalization bounds for some kernel learning problems. The resulting convergence rates are optimal for various learning problems. We have also shown that the

effective dimension at the appropriate chosen optimal scale can be sample-size dependent and behaves like $\sqrt{n}$ in the worst case.

This shows that despite the claim that a kernel method learns a predictor from an infinite dimensional Hilbert space, for a fixed sample size, the effective dimension is rather small. This in fact indicates that they are not any more powerful than learning in an appropriately chosen finite dimensional space. This observation also raises the following computational question: given $n$-samples, kernel methods use $n$ parameters in the computation but as we have shown, the effective number of parameters (effective dimension) is not more than $O(\sqrt{n})$. Therefore it could be possible to significantly reduce the computational cost of kernel methods by explicitly parameterizing the effective dimensions.

## A    Properties of scale-sensitive effective data dimension

We list some properties of the scale-sensitive data dimension $D_\lambda$. Due to the limitation of space, we shall skip the proofs. The following lemma implies that the quantity $D_\lambda$ behaves like dimension if the underlying space $H$ is finite dimensional.

**Proposition A.1** *If $H$ is a finite dimensional space, then $D_\lambda \leq \dim(H)$. Moreover, for all Hilbert spaces $H$, we have the following bound $D_\lambda \leq M_H^2/\lambda$, where $M_H$ is defined in (5).*

**Proposition A.2** *Consider the complete set of ortho-normal eigen-pairs $\{(\lambda_i, u_i) : i \geq 1\}$ of the operator $E_x \phi_x \phi_x^T$, where $u_i \cdot u_j = 0$ if $i \neq j$ and $u_i \cdot u_i = 1$. This gives the decomposition: $E_x \phi_x \phi_x^T = \sum_i \lambda_i u_i u_i^T$, where $\lambda_i = E_x u_i(x)^2$. We have the identity: $D_\lambda = \sum_i \frac{\lambda_i}{\lambda_i + \lambda}$.*

In many cases, we can find a so-called feature representation of the kernel function $K(x_1, x_2) = \phi_{x_1} \cdot \phi_{x_2}$. In such cases the eigenvalues $\lambda_k$ can be easily bounded.

**Proposition A.3** *Consider the following feature space decomposition of kernel: $\phi_{x_1} \cdot \phi_{x_2} = \sum_i \psi_i(x_1)\psi_i(x_2)$, where each $\psi_i$ is a real valued function. If $\lambda_1 \geq \lambda_2 \cdots$, then we have the following bound: $\sum_{i \geq k} \lambda_i \leq E_x \sum_{i \geq k} \psi_i(x)^2$. This implies*

$$ D_\lambda \leq \inf_{k \geq 0} \left[ k + \sup_x \sum_{i > k} \psi_i(x)^2/\lambda \right]. $$

## Footnotes

[1]This boundedness assumption is not essential. However in this paper, in order to emphasize the main idea, we shall avoid using a more complex derivation that handles more general situations.

[2]The lower bound is well-known in the non-parametric statistical literature (for example, see [3]).

## References

[1] W.S. Lee, P.L. Bartlett, and R.C. Williamson. The importance of convexity in learning with squared loss. *IEEE Trans. Inform. Theory*, 44(5):1974–1980, 1998.

[2] Shahar Mendelson. Learning relatively small classes. In *COLT 01*, pages 273–288, 2001.

[3] Charles J. Stone. Optimal global rates of convergence for nonparametric regression. *Annals of Statistics*, 10:1040–1053, 1982.

[4] S.A. van de Geer. *Empirical Processes in $M$-estimation*. Cambridge University Press, 2000.

[5] Vadim Yurinsky. *Sums and Gaussian vectors*. Springer-Verlag, Berlin, 1995.
